# Neural Decoding of Cursor Motion Using a Kalman Filter

**W. Wu**[*]    **M. J. Black**[†]    **Y. Gao**[*]    **E. Bienenstock**[*§]
**M. Serruya**[§¶]    **A. Shaikhouni**[§¶]    **J. P. Donoghue**[§]

[*]Division of Applied Mathematics, [†]Dept. of Computer Science,
[§]Dept. of Neuroscience, [¶]Division of Biology and Medicine,
Brown University, Providence, RI 02912

`weiwu@cfm.brown.edu`, `black@cs.brown.edu`, `gao@cfm.brown.edu`,
`elie@dam.brown.edu`, `Mijail_Serruya@brown.edu`,
`Ammar_Shaikhouni@brown.edu`, `john_donoghue@brown.edu`

## Abstract

The direct neural control of external devices such as computer displays
or prosthetic limbs requires the accurate decoding of neural activity rep-
resenting continuous movement. We develop a real-time control system
using the spiking activity of approximately 40 neurons recorded with
an electrode array implanted in the arm area of primary motor cortex.
In contrast to previous work, we develop a control-theoretic approach
that explicitly models the motion of the hand and the probabilistic re-
lationship between this motion and the mean firing rates of the cells in
$70ms$ bins. We focus on a realistic cursor control task in which the sub-
ject must move a cursor to "hit" randomly placed targets on a computer
monitor. Encoding and decoding of the neural data is achieved with a
Kalman filter which has a number of advantages over previous linear
filtering techniques. In particular, the Kalman filter reconstructions of
hand trajectories in off-line experiments are more accurate than previ-
ously reported results and the model provides insights into the nature of
the neural coding of movement.

## 1   Introduction

Recent results have demonstrated the feasibility of direct neural control of devices such as
computer cursors using implanted electrodes [5, 9, 11, 14]. These results are enabled by a
variety of mathematical "decoding" methods that produce an estimate of the system "state"
(e.g. hand position) from a sequence of measurements (e.g. the firing rates of a collection
of cells). Here we argue that such a decoding method should (1) have a sound probabilistic
foundation; (2) explicitly model noise in the data; (3) indicate the uncertainty in estimates
of hand position; (4) make minimal assumptions about the data; (5) require a minimal
amount of "training" data; (6) provide on-line estimates of hand position with short delay
(less than 200ms); and (7) provide insight into the neural coding of movement. To that

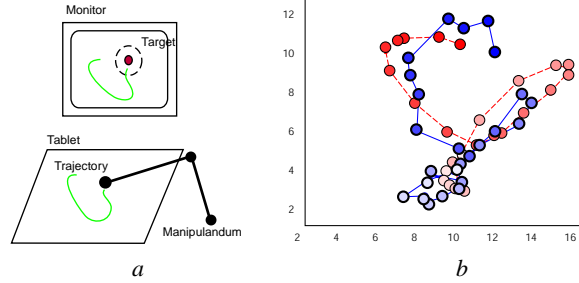

*a*                                  *b*

Figure 1: Reconstructing 2D hand motion. *(a)* **Training:** neural spiking activity is recorded while the subject moves a jointed manipulandum on a 2D plane to control a cursor so that it hits randomly placed targets. *(b)* **Decoding:** true target trajectory (dashed (red): dark to light) and reconstruction using the Kalman filter (solid (blue): dark to light).

end, we propose a Kalman filtering method that provides a rigorous and well understood framework that addresses these issues. This approach provides a control-theoretic model for the encoding of hand movement in motor cortex and for inferring, or decoding, this movement from the firing rates of a population of cells.

Simultaneous recordings are acquired from an array consisting of 100 microelectrodes [6] implanted in the arm area of primary motor cortex (MI) of a Macaque monkey; recordings from this area have been used previously to control devices [5, 9, 10, 11, 14]. The monkey views a computer monitor while gripping a two-link manipulandum that controls the 2D motion of a cursor on the monitor (Figure 1*a*). We use the experimental paradigm of [9], in which a target dot appears in a random location on the monitor and the task requires moving a feedback dot with the manipulandum so that it hits the target. When the target is hit, it jumps to a new random location. The trajectory of the hand and the neural activity of 42 cells are recorded simultaneously. We compute the position, velocity, and acceleration of the hand along with the mean firing rate for each of the cells within non-overlapping $70ms$ time bins. In contrast to related work [8, 15], the motions of the monkey in this task are quite rapid and more "natural" in that the actual trajectory of the motion is unconstrained.

The reconstruction of hand trajectory from the mean firing rates can be viewed probabilistically as a problem of *inferring* behavior from noisy measurements. In [15] we proposed a Kalman filter framework [3] for modeling the relationship between firing rates in motor cortex and the position and velocity of the subject's hand. This work focused on off-line reconstruction using constrained motions of the hand [8]. Here we consider new data from the on-line environmental setup [9] which is more natural, varied, and contains rapid motions. With this data we show that, in contrast to our previous results, a model of hand acceleration (in addition to position and velocity) is important for accurate reconstruction.

In the Kalman framework, the hand movement (position, velocity and acceleration) is modeled as the system *state* and the neural firing rate is modeled as the *observation* (measurement). The approach specifies an explicit generative model that assumes the observation (firing rate in $70ms$) is a linear function of the state (hand kinematics) plus Gaussian noise[1]. Similarly, the hand state at time $t$ is assumed to be a linear function of the hand state at the previous time instant plus Gaussian noise. The Kalman filter approach provides a recursive, on-line, estimate of hand kinematics from the firing rate in non-overlapping time bins. The

results of reconstructing hand trajectories from pre-recorded neural firing rates are compared with those obtained using more traditional fixed linear filtering techniques [9, 12] using overlapping $1.4s$ windows. The results indicate that the Kalman filter decoding is more accurate than that of the fixed linear filter.

## 1.1 Related Work

Georgopoulos and colleagues [4] showed that hand movement direction may be encoded by the neural ensemble in the arm area of motor cortex (MI). This early work has resulted in a number of successful algorithms for decoding neural activity in MI to perform off-line reconstruction or on-line control of cursors or robotic arms. Roughly, the primary methods for decoding MI activity include the *population vector* algorithm [4, 5, 7, 11], linear filtering [9, 12], artificial neural networks [14], and probabilistic methods [2, 10, 15].

This population vector approach is the oldest method and it has been used for the real-time neural control of 3D cursor movement [11]. This work has focused primarily on "center out" motions to a discrete set of radial targets (in 2D or 3D) rather than natural, continuous, motion that we address here.

Linear filtering [8, 12] is a simple statistical method that is effective for real-time neural control of a 2D cursor [9]. This approach requires the use of data over a long time window (typically $500ms$ to $1.5s$). The fixed linear filter, like population vectors and neural networks [14] lack both a clear probabilistic model and a model of the temporal hand kinematics. Additionally, they provide no estimate of uncertainty and hence may be difficult to extend to the analysis of more complex temporal movement patterns.

We argue that what is needed is a probabilistically grounded method that uses data in small time windows (e.g. $50 - 100ms$ or less) and integrates that information over time in a recursive fashion. The CONDENSATION algorithm has been recently introduced as a Bayesian decoding scheme [2], which provides a probabilistic framework for causal estimation and is shown superior to the performance of linear filtering when sufficient data is available (e.g. using firing rates for several hundred cells). Note that the CONDENSATION method is more general than the Kalman filter proposed here in that it does not assume linear models and Gaussian noise. While this may be important for neural decoding as suggested in [2], current technology makes the method impractical for real-time control.

For real-time neural control we exploit the Kalman filter [3, 13] which has been widely used for estimation problems ranging from target tracking to vehicle control. Here we apply this well understood theory to the problem of decoding hand kinematics from neural activity in motor cortex. This builds on the work that uses recursive Bayesian filters to estimate the position of a rat from the firing activity of hippocampal place cells [1, 16]. In contrast to the linear filter or population vector methods, this approach provides a measure of confidence in the resulting estimates. This can be extremely important when the output of the decoding method is to be used for later stages of analysis.

## 2 Methods

Decoding involves estimating the *state* of the hand at the current instant in time; i.e. $\mathbf{x}_k = [x, y, v_x, v_y, a_x, a_y]_k^T$ representing $x$-position, $y$-position, $x$-velocity, $y$-velocity, $x$-acceleration, and $y$-acceleration at time $t_k = k\Delta t$ where $\Delta t = 70ms$ in our experiments. The Kalman filter [3, 13] model assumes the state is linearly related to the observations $\mathbf{z}_k \in \Re^C$ which here represents a $C \times 1$ vector containing the firing rates at time $t_k$ for $C$

observed neurons within $70ms$. In our experiments, $C = 42$ cells. We briefly review the Kalman filter algorithm below; for details the reader is referred to [3, 13].

**Encoding:** We define a *generative model* of neural firing as

$$\mathbf{z}_k = \mathbf{H}_k \mathbf{x}_k + \mathbf{q}_k, \tag{1}$$

where $k = 1, 2, \cdots, M$, $M$ is the number of time steps in the trial, and $\mathbf{H} \in \Re^{C \times 6}$ is a matrix that linearly relates the hand state to the neural firing. We assume the noise in the observations is zero mean and normally distributed, i.e. $\mathbf{q}_k \sim N(0, \mathbf{Q}_k), \mathbf{Q}_k \in \Re^{C \times C}$.

The states are assumed to propagate in time according to the system model

$$\mathbf{x}_{k+1} = \mathbf{A}_k \mathbf{x}_k + \mathbf{w}_k, \tag{2}$$

where $\mathbf{A}_k \in \Re^{6 \times 6}$ is the coefficient matrix and the noise term $\mathbf{w}_k \sim N(0, \mathbf{W}_k), \mathbf{W}_k \in \Re^{6 \times 6}$. This states that the hand kinematics (position, velocity, and acceleration) at time $k + 1$ is linearly related to the state at time $k$. Once again we assume these estimates are normally distributed.

In practice, $\mathbf{A}_k, \mathbf{H}_k, \mathbf{W}_k, \mathbf{Q}_k$ might change with time step $k$, however, here we make the common simplifying assumption they are constant. Thus we can estimate the Kalman filter model from training data using least squares estimation:

$$\operatorname*{argmin}_{\mathbf{A}} \sum_{k=1}^{M-1} ||\mathbf{x}_{k+1} - \mathbf{A}\mathbf{x}_k||^2, \quad \operatorname*{argmin}_{\mathbf{H}} \sum_{k=1}^{M} ||\mathbf{z}_k - \mathbf{H}\mathbf{x}_k||^2,$$

where $||\cdot||$ is the $\mathbf{L}^2$ norm. Given $\mathbf{A}$ and $\mathbf{H}$ it is then simple to estimate the noise covariance matrices $\mathbf{W}$ and $\mathbf{Q}$; details are given in [15].

**Decoding:** At each time step $k$ the algorithm has two steps: 1) prediction of the *a priori* state estimate $\hat{\mathbf{x}}_k^-$; and 2) updating this estimate with new measurement data to produce an *a posteriori* state estimate $\hat{\mathbf{x}}_k$. In particular, these steps are:

**I. Discrete Kalman filter time update equations:**

At each time $t_k$, we obtain the *a priori* estimate from the previous time $t_{k-1}$, then compute its error covariance matrix, $\mathbf{P}_k^-$:

$$\hat{\mathbf{x}}_k^- = \mathbf{A}\hat{\mathbf{x}}_{k-1}, \tag{3}$$

$$\mathbf{P}_k^- = \mathbf{A}\mathbf{P}_{k-1}\mathbf{A}^T + \mathbf{W}. \tag{4}$$

**II. Measurement update equations:**

Using the estimate $\hat{\mathbf{x}}_k^-$ and firing rate $\mathbf{z}_k$, we update the estimate using the measurement and compute the posterior error covariance matrix:

$$\hat{\mathbf{x}}_k = \hat{\mathbf{x}}_k^- + \mathbf{K}_k(\mathbf{z}_k - \mathbf{H}\hat{\mathbf{x}}_k^-), \tag{5}$$

$$\mathbf{P}_k = (\mathbf{I} - \mathbf{K}_k\mathbf{H})\mathbf{P}_k^-, \tag{6}$$

where $\mathbf{P}_k$ represents the state error covariance after taking into account the neural data and $\mathbf{K}_k$ is the Kalman *gain* matrix given by

$$\mathbf{K}_k = \mathbf{P}_k^- \mathbf{H}^T(\mathbf{H}\mathbf{P}_k^- \mathbf{H}^T + \mathbf{Q})^{-1}. \tag{7}$$

This $\mathbf{K}_k$ produces a state estimate that minimizes the mean squared error of the reconstruction (see [3] for details). Note that $\mathbf{Q}$ is the measurement error matrix and, depending on the reliability of the data, the gain term, $\mathbf{K}_k$, automatically adjusts the contribution of the new measurement to the state estimate.

| Method | Correlation Coefficient $(x, y)$ | MSE $(cm^2)$ |
|---|---|---|
| Kalman ($0ms$ lag) | $(0.768, 0.912)$ | 7.09 |
| Kalman ($70ms$ lag) | $(0.785, 0.932)$ | 7.07 |
| **Kalman ($140ms$ lag)** | **$(0.815, 0.929)$** | **6.28** |
| Kalman ($210ms$ lag) | $(0.808, 0.891)$ | 6.87 |
| Kalman (no acceleration) | $(0.817, 0.914)$ | 6.60 |
| Linear filter | $(0.756, 0.915)$ | 8.30 |

Table 1: Reconstruction results for the fixed linear and recursive Kalman filter. The table also shows how the Kalman filter results vary with lag times (see text).

## 3   Experimental Results

To be practical, we must be able to train the model (i.e. estimate $\mathbf{A}$, $\mathbf{H}$, $\mathbf{W}$, $\mathbf{Q}$) using a small amount of data. Experimentally we found that approximately 3.5 minutes of training data suffices for accurate reconstruction (this is similar to the result for fixed linear filters reported in [9]). As described in the introduction, the task involves moving a manipulandum freely on a $30cm \times 30cm$ tablet (with a $20cm \times 20cm$ workspace) to hit randomly placed targets on the screen. We gather the mean firing rates and actual hand trajectories for the training data and then learn the models via least squares (the computation time is negligible). We then test the accuracy of the method by reconstructing test trajectories off-line using recorded neural data not present in the training set. The results reported here use approximately 1 minute of test data.

**Optimal Lag:** The physical relationship between neural firing and arm movement means there exists a time lag between them [7, 8]. The introduction of a time lag results in the measurements, $\mathbf{z}_k$, at time $t_k$, being taken from some previous (or future) instant in time $t_{k-i}$ for some integer $i$. In the interest of simplicity, we consider a single optimal time lag for all the cells though evidence suggests that individual time lags may provide better results [15].

Using time lags of 0, 70, 140, 210 $ms$ we train the Kalman filter and perform reconstruction (see Table 1). We report the accuracy of the reconstructions with a variety of error measures used in the literature including the correlation coefficient ($r$) and the mean squared error (MSE) between the reconstructed and true trajectories. From Table 1 we see that optimal lag is around two time steps (or $140ms$); this lag will be used in the remainder of the experiments and is similar to our previous findings [15] which suggested that the optimal lag was between 50-100$ms$.

**Decoding:** At the beginning of the test trial we let the predicted initial condition equal the real initial condition. Then the update equations in Section 2 are applied. Some examples of the reconstructed trajectory are shown in Figure 2 while Figure 3 shows the reconstruction of each component of the state variable (position, velocity and acceleration in $x$ and $y$).

From Figure 3 and Table 1 we note that the reconstruction in $y$ is more accurate than in the $x$ direction (the same is true for the fixed linear filter described below); this requires further investigation. Note also that the ground truth velocity and acceleration curves are computed from the position data with simple differencing. As a result these plots are quite noisy making an evaluation of the reconstruction difficult.

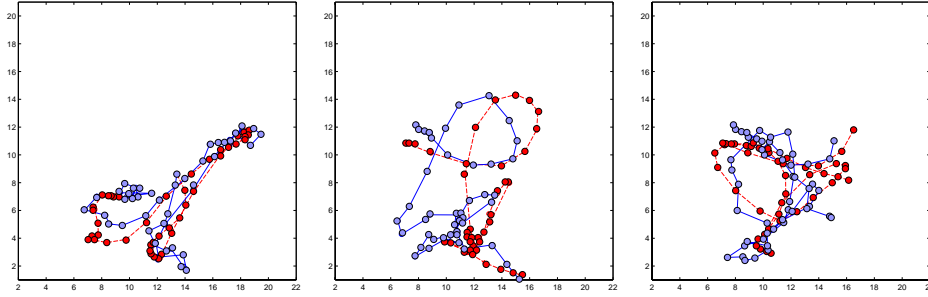

Figure 2: Reconstructed trajectories (portions of $1min$ test data – each plot shows 50 time instants ($3.5s$)): true target trajectory (dashed (red)) and reconstruction using the Kalman filter (solid (blue)).

### 3.1 Comparison with linear filtering

Fixed linear filters reconstruct hand position as a linear combination of the firing rates over some fixed time period [4, 9, 12]; that is,

$$x_k = a + \sum_v \sum_{j=0}^{N} r_{k-j}^v f_j^v,$$

where $x_k$ is the $x$-position (or, equivalently, the $y$-position) at time $t_k = k\Delta t$ ($\Delta t = 70ms$), $k = 1, \cdots, M$, where $M$ is the number of time steps in a trial, $a$ is the constant offset, $r_{k-j}^v$ is the firing rate of neuron $v$ at time $t_{k-j}$, and $f_j^v$ are the filter coefficients. The coefficients can be learned from training data using a simple least squares technique. In our experiments here we take $N = 20$ which means that the hand position is determined from firing data over $1.4s$. This is exactly the method described in [9] which provides a fair comparison for the Kalman filter; for details see [12, 15]. Note that since the linear filter uses data over a long time window, it does not benefit from the use of time-lagged data. Note also that it does not explicitly reconstruct velocity or acceleration.

The linear filter reconstruction of position is shown in Figure 4. Compared with Figure 3, we see that the results are visually similar. Table 1, however, shows that the Kalman filter gives a more accurate reconstruction than the linear filter (higher correlation coefficient and lower mean-squared error). While fixed linear filtering is extremely simple, it lacks many of the desirable properties of the Kalman filter.

**Analysis:** In our previous work [15], the experimental paradigm involved carefully designed hand motions that were slow and smooth. In that case we showed that acceleration was redundant and could be removed from the state equation. The data used here is more "natural", varied, and rapid and we find that modeling acceleration improves the prediction of the system state and the accuracy of the reconstruction; Table 1 shows the decrease in accuracy with only position and velocity in the system state (with $140ms$ lag).

## 4 Conclusions

We have described a discrete linear Kalman filter that is appropriate for the neural control of 2D cursor motion. The model can be easily learned using a few minutes of training data and provides real-time estimates of hand position every $70ms$ given the firing rates of 42

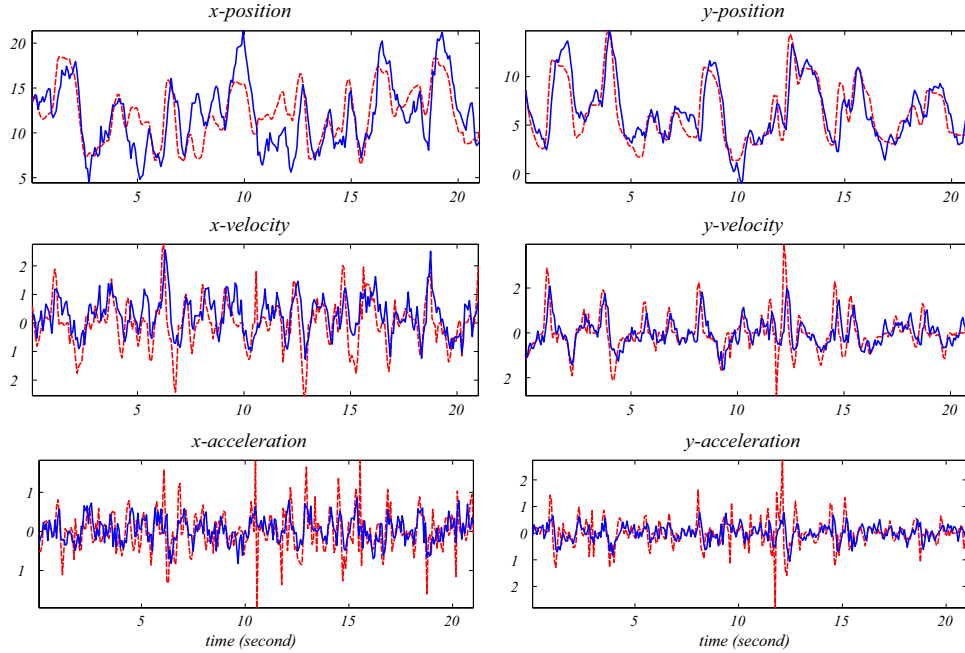

Figure 3: Reconstruction of each component of the system state variable: true target motion (dashed (red)) and reconstruction using the Kalman filter (solid (blue)). $20s$ from a $1min$ test sequence are shown.

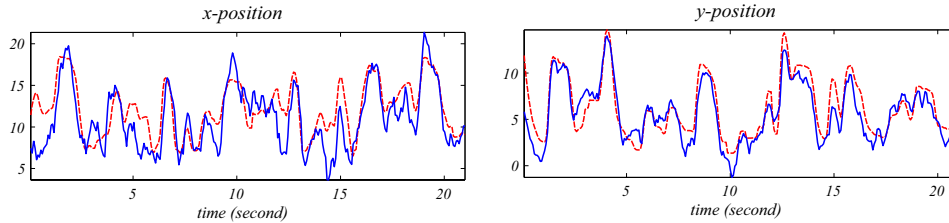

Figure 4: Reconstruction of position using the linear filter: true target trajectory (dashed (red)) and reconstruction using the linear filter (solid (blue)).

cells in primary motor cortex. The estimated trajectories are more accurate than the fixed linear filtering results being used currently.

The Kalman filter proposed here provides a rigorous probabilistic approach with a well understood theory. By making its assumptions explicit and by providing an estimate of uncertainty, the Kalman filter offers significant advantages over previous methods. The method also estimates hand velocity and acceleration in addition to 2D position. In contrast to previous experiments, we show, for the natural 2D motions in this task, that incorporating acceleration into the system and measurement models improves the accuracy of the decoding. We also show that, consistent with previous studies, a time lag of $70 - 140ms$ improves the accuracy.

Our future work will evaluate the performance of the Kalman filter for on-line neural control of cursor motion in the task described here. Additionally, we are exploring alternative measurement noise models, non-linear system models, and non-linear particle filter decod-

ing methods. Finally, to get a complete picture of current methods, we are pursuing further comparisons with population vector methods [7] and particle filtering techniques [2].

**Acknowledgments.** This work was supported in part by: the DARPA Brain Machine Interface Program, NINDS Neural Prosthetics Program and Grant #NS25074, and the National Science Foundation (ITR Program award #0113679). We thank J. Dushanova, C. Vargas, L. Lennox, and M. Fellows for their assistance.

## Footnotes

[1]This is a crude assumption but the firing rates can be square-root transformed [7] making them more Gaussian and the mean firing rate can be subtracted to achieve zero-mean data.

# References

[1] Brown, E., Frank, L., Tang, D., Quirk, M., and Wilson, M. (1998). A statistical paradigm for neural spike train decoding applied to position prediction from ensemble firing patterns of rat hippocampal place cells. *J. of Neuroscience*, 18(18):7411–7425.

[2] Gao, Y., Black, M. J., Bienenstock, E., Shoham, S., and Donoghue, J. P. (2002). Probabilistic inference of hand motion from neural activity in motor cortex. *Advances in Neural Information Processing Systems 14, The MIT Press*.

[3] Gelb, A., (Ed.) (1974). *Applied Optimal Estimation*. MIT Press.

[4] Georgopoulos, A., Schwartz, A., and Kettner, R. (1986). Neural population coding of movement direction. *Science*, 233:1416–1419.

[5] Helms Tillery, S., Taylor, D., Isaacs, R., Schwartz, A. (2000) Online control of a prosthetic arm from motor cortical signals. *Soc. for Neuroscience Abst.*, Vol. 26.

[6] Maynard, E., Nordhausen C., Normann, R. (1997). The Utah intracortical electrode array: A recording structure for potential brain-computer interfaces. *Electroencephalography and Clinical Neuophysiology* 102, pp. 228–239.

[7] Moran, D. and Schwartz, B. (1999). Motor cortical representation of speed and direction during reaching. *J. of Neurophysiology*, 82(5):2676–2692.

[8] Paninski, L., Fellows, M., Hatsopoulos, N., and Donoghue, J. P. (2001). Temporal tuning properties for hand position and velocity in motor cortical neurons. *submitted, J. of Neurophysiology*.

[9] Serruya, M. D., Hatsopoulos, N. G., Paninski, L., Fellows, M. R., and Donoghue, J. P. (2002). Brain-machine interface: Instant neural control of a movement signal. *Nature*, (416):141–142.

[10] Serruya. M., Hatsopoulos, N., Donoghue, J., (2000) Assignment of primate M1 cortical activity to robot arm position with Bayesian reconstruction algorithm. *Soc. for Neuro. Abst.*, Vol. 26.

[11] Taylor. D., Tillery, S., Schwartz, A. (2002). Direct cortical control of 3D neuroprosthetic devices. *Science*, Jun. 7;296(5574):1829-32.

[12] Warland, D., Reinagel, P., and Meister, M. (1997). Decoding visual information from a population of retinal ganglion cells. *J. of Neurophysiology*, 78(5):2336–2350.

[13] Welch, G. and Bishop, G. (2001). An introduction to the Kalman filter. Technical Report TR 95-041, University of North Carolina at Chapel Hill, Chapel Hill,NC 27599-3175.

[14] Wessberg, J., Stambaugh, C., Kralik, J., Beck, P., Laubach, M., Chapin, J., Kim, J., Biggs, S., Srinivasan, M., and Nicolelis, M. (2000). Real-time prediction of hand trajectory by ensembles of cortical neurons in primates. *Nature*, 408:361–365.

[15] Wu, W., Black, M. J., Gao, Y., Bienenstock, E., Serruya, M., and Donoghue, J. P., Inferring hand motion from multi-cell recordings in motor cortex using a Kalman filter, *SAB'02-Workshop on Motor Control in Humans and Robots: On the Interplay of Real Brains and Artificial Devices*, Aug. 10, 2002, Edinburgh, Scotland, pp. 66–73.

[16] Zhang, K., Ginzburg, I., McNaughton, B., Sejnowski, T., Interpreting neuronal population activity by reconstruction: Unified framework with application to hippocampal place cells, *J. Neurophysiol.* 79:1017–1044, 1998.
